# Information Regularization with Partially Labeled Data

**Martin Szummer**
MIT AI Lab & CBCL
Cambridge, MA 02139
*szummer@ai.mit.edu*

**Tommi Jaakkola**
MIT AI Lab
Cambridge, MA 02139
*tommi@ai.mit.edu*

## Abstract

Classification with partially labeled data requires using a large number of unlabeled examples (or an estimated marginal $P(\boldsymbol{x})$), to further constrain the conditional $P(y|\boldsymbol{x})$ beyond a few available labeled examples. We formulate a regularization approach to linking the marginal and the conditional in a general way. The regularization penalty measures the information that is implied about the labels over covering regions. No parametric assumptions are required and the approach remains tractable even for continuous marginal densities $P(\boldsymbol{x})$. We develop algorithms for solving the regularization problem for finite covers, establish a limiting differential equation, and exemplify the behavior of the new regularization approach in simple cases.

## 1 Introduction

Many modern classification problems are rife with unlabeled examples. To benefit from such examples, we must exploit either implicitly or explicitly the link between the marginal density $P(\boldsymbol{x})$ over examples $\boldsymbol{x}$ and the conditional $P(y|\boldsymbol{x})$ representing the decision boundary for the labels $y$. High density regions or clusters in the data, for example, can be expected to fall solely in one or another class.

Most discriminative methods do not attempt to explicitly model or incorporate information from the marginal density $P(\boldsymbol{x})$. However, many discriminative algorithms such as SVMs exploit the notion of margin that effectively relates $P(\boldsymbol{x})$ to $P(y|\boldsymbol{x})$; the decision boundary is biased to fall preferentially in low density regions of $P(\boldsymbol{x})$ so that only a few points fall within the margin band.

The assumptions relating $P(\boldsymbol{x})$ to $P(y|\boldsymbol{x})$ are seldom made explicit. In this paper we appeal to information theory to explicitly constrain $P(y|\boldsymbol{x})$ on the basis of $P(\boldsymbol{x})$ in a regularization framework. The idea is in broad terms related to a number of previous approaches including maximum entropy discrimination [1], data clustering by information bottleneck [2], and minimum entropy data partitioning [3]. See also [4].

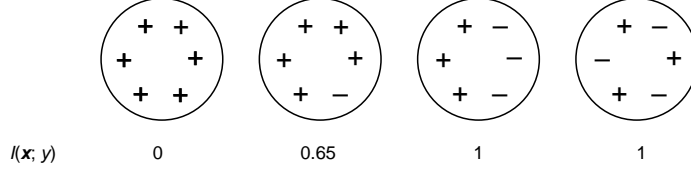

$I(\boldsymbol{x}; y)$        0          0.65          1          1

Figure 1: Mutual information $I(\boldsymbol{x}; y)$ measured in bits for four regions with different configurations of labels $y=\{+,-\}$. The marginal $P(\boldsymbol{x})$ is discrete and uniform across the points. The mutual information is low when the labels are homogenous in the region, and high when labels vary. The mutual information is invariant to the spatial configuration of points within the neighborhood.

## 2 Information Regularization

We begin by showing how to regularize a small region of the domain $\mathcal{X}$. We will subsequently cover the domain (or any chosen subset) with multiple small regions, and describe criteria that ensure regularization of the whole domain on the basis of the individual regions.

### 2.1 Regularizing a Single Region

Consider a small contiguous region $Q$ in the domain $\mathcal{X}$ (e.g., an $\epsilon$-ball). We will regularize the conditional probability $P(y|\boldsymbol{x})$ by penalizing the amount of information the conditionals imply about the labels within the region.

The regularizer is a function of both $P(y|\boldsymbol{x})$ and $P(\boldsymbol{x})$, and will penalize changes in $P(y|\boldsymbol{x})$ more in regions with high $P(\boldsymbol{x})$. Let $L$ be the set of labeled points (size $N_L$) and $L \cup U$ be the set of labeled and unlabeled points (size $N_{LU}$). The marginal $P(\boldsymbol{x})$ is assumed to be given, and may be available directly in terms of a continuous density, or as an empirical density $P(\boldsymbol{x}) = 1/N_{LU} \cdot \sum_{i \in L \cup U} \delta(\boldsymbol{x} - \boldsymbol{x}_i)$ corresponding to a set of points $\{\boldsymbol{x}_i\}$ that may not have labels ($\delta(\cdot)$ is the Dirac delta function integrating to 1).

As a measure of information, we employ mutual information [5], which is the average number of bits that $\boldsymbol{x}$ contains about the label in region $Q$ (see Figure 1.) The measure depends both on the marginal density $P(\boldsymbol{x})$ (specifically its restriction to $\boldsymbol{x} \in Q$ namely $P(\boldsymbol{x}|Q) = P(\boldsymbol{x})/\int_Q P(\boldsymbol{x})\,\mathrm{d}\boldsymbol{x}$) and the conditional $P(y|\boldsymbol{x})$. Equivalently, we can interpret mutual information as a measure of disagreement among $P(y|\boldsymbol{x})$, $\boldsymbol{x} \in Q$. The measure is zero for any constant $P(y|\boldsymbol{x})$. More precisely, the mutual information in region $Q$ is

$$I_Q(\boldsymbol{x}; y) = \sum_y \int_{\boldsymbol{x} \in Q} P(\boldsymbol{x}|Q)P(y|\boldsymbol{x}) \log \frac{P(y|\boldsymbol{x})}{P(y|Q)}\,\mathrm{d}\boldsymbol{x}, \tag{1}$$

where $P(y|Q) = \int_{\boldsymbol{x} \in Q} P(\boldsymbol{x}|Q)P(y|\boldsymbol{x})\,\mathrm{d}\boldsymbol{x}$. The densities conditioned on $Q$ are normalized to integrate to 1 within the region $Q$. Note that the mutual information is invariant to permutations of the elements of $\mathcal{X}$ within $Q$, which suggests that the regions must be small enough to preserve locality.

The regularization penalty has to further scale with the number of points in the region (or the probability mass). We introduce the following regularization principle:

**Information regularization**
penalize $(M_Q/V_Q) \cdot I_Q(\boldsymbol{x}; y)$, which is the information about the labels within a local region $Q$, weighted by the overall probability mass $M_Q$ in the region, and normalized by a measure of variability $V_Q$ (variance) of $\boldsymbol{x}$ in the region.

Here $M_Q = \int_{\boldsymbol{x} \in Q} P(\boldsymbol{x}) \, d\boldsymbol{x}$. The mutual information $I_Q(\boldsymbol{x}; y)$ measures the information *per point*, and to obtain the total mutual information contained in a region, we must multiply by the probability mass $M_Q$. The regularization will be stronger in regions with high $P(\boldsymbol{x})$.

$V_Q$ is a measure of variance of $\boldsymbol{x}$ restricted to the region, and is introduced to remove overall dependence on the size of the region. In one dimension, $V_Q = \text{var}(x|Q)$. When the region is small, then the marginal will be close to uniform over the region and $V_Q \propto R^2$, where $R$ is, e.g., the radius for spherical regions. We omit here the analysis of the $d$-dimensional case and only note that we may choose $V_Q = \text{tr} \, \Sigma_Q$, where the covariance $\Sigma_Q = \int_{x \in Q} (\boldsymbol{x} - E_Q(\boldsymbol{x}))(\boldsymbol{x} - E_Q(\boldsymbol{x}))^T P(\boldsymbol{x}|Q) \, d\boldsymbol{x}$. The choice of $V_Q$ is based on the limiting argument discussed next.

## 2.2 Limiting Behavior for Vanishing Size Regions

When the size of the region is scaled down, the mutual information will go to zero for any continuous $P(y|\boldsymbol{x})$. We derive here the appropriate regularization penalty in the limit of vanishing regions. For simplicity, we only consider the one-dimensional case.

Within a small region $Q$ we can (under mild continuity assumptions) approximate $P(y|x)$ by a Taylor expansion around the mean point $x_0 \in Q$, obtaining $P(y|Q) \approx P(y|x_0)$ to first order. By using $\log(1 + z) \approx z - z^2/2$ and substituting the approximate $P(y|x)$ and $P(y|Q)$ into $I_Q(x; y)$, we get the following first order expression for mutual information:

$$I_Q(x; y) = \frac{1}{2} \underbrace{\text{var}(x|Q)}_{\text{size-dependent}} \underbrace{\sum_y P(y|x_0) \left. \frac{d \log P(y|x)}{dx} \right|_{x_0}^2}_{\text{size-independent}} \tag{2}$$

$\text{var}(x|Q)$ is dependent on the size (and more generally shape) of region $Q$ while the remaining parts are independent of the size (and shape). The regularization penalty should not scale with the resolution at which we penalize information and we thus divide out the size-dependent part.

The size-independent part is the Fisher information [5], where we think of $P(y|x)$ as parameterized by $x$. The expression $d \log P(y|x)/dx$ is known as the Fisher score.

## 2.3 Regularizing the Domain

We want to regularize the conditional $P(y|\boldsymbol{x})$ across the domain $\mathcal{X}$ (or any subset of interest). Since individual regions must be relatively small to preserve locality, we need multiple regions to cover the domain. The cover is the set $\mathcal{C}$ of these regions. Since the regularization penalty is assigned to each region, the regions must overlap to ensure that the conditionals in different regions become functionally dependent. See Figure 2.

In general all areas with significant marginal density $P(\boldsymbol{x})$ should be included in the cover or will not be regularized (areas of zero marginal need not be considered). The cover should generally be connected (with respect to neighborhood relations of the regions) so that labeled points have potential to influence all conditionals. The amount of overlap between any two regions in the cover determines how strongly the corresponding conditionals are

tied to each other. On the other hand, the regions should be small to preserve locality. The limit of a large number of small overlapping regions can be defined, and we ensure continuity of $P(y|\boldsymbol{x})$ when the offset between regions vanishes relative to their size (in all dimensions).

# 3 Classification with Information Regularization

Information regularization across multiple regions can be performed, for example, by minimizing the maximum information per region, subject to correct classification of the labeled points. Specifically, we constrain each region in the cover ($Q \in \mathcal{C}$) to carry at most $\gamma$ units of information.

$$
\begin{array}{lll}
\min_{P(y|\boldsymbol{x}_k),\,\gamma} & \gamma & \text{(3a)} \\
\text{s.t.} & (M_Q/V_Q) \cdot I_Q(\boldsymbol{x};y) \leq \gamma & \forall Q \in \mathcal{C} \quad \text{(3b)} \\
& P(y|\boldsymbol{x}_k) = \delta(y,\tilde{y}_k) & \forall k \in L \quad \text{(3c)} \\
& 0 \leq P(y|\boldsymbol{x}_k) \leq 1, \quad \sum_y P(y|\boldsymbol{x}_k) = 1 & \forall k \in L \cup U,\ \forall y. \quad \text{(3d)}
\end{array}
$$

We have incorporated the labeled points by constraining their conditionals to the observed values (eq. 3c) (see below for other ways of incorporating labeled information). The solution $P(y|\boldsymbol{x})$ to this optimization problem is unique in regions that achieve the information constraint with equality (as long as $P(\boldsymbol{x}) > 0$). (Uniqueness follows from the strict convexity of mutual information as a function of $P(y|\boldsymbol{x})$ for nonzero $P(\boldsymbol{x})$).

Define an *atomic* subregion as a non-empty intersection of regions that cannot be further intersected by any region (Figure 2). All unlabeled points in an atomic subregion belong to the same set of regions, and therefore participate in exactly the same constraints. They will be regularized the same way, and since mutual information is a convex function, it will be minimized when the conditionals $P(y|\boldsymbol{x})$ are equal in the atomic subregion. We can therefore parsimoniously represent conditionals of atomic subregions, instead of individual points, merely by treating such atomic subregions as "merged points" and weighting the associated constraint by the probability mass contained in the subregion.

## 3.1 Incorporating Noisy Labels

Labeled points participate in the information regularization in the same way as unlabeled points. However, their conditionals have additional constraints, which incorporate the label information. In equation 3c we used the constraint $P(y|\boldsymbol{x}_k) = \delta(y,\tilde{y}_k)$ for all labeled points. This constraint does not permit noise in the labels (and cannot be used when two points at the same location have disagreeing labels.) Alternatively, we can apply either of the constraints

(`fix-lbl`): $P(y|\boldsymbol{x}_i) = (1-b)^{\delta(y,\tilde{y}_i)} b^{1-\delta(y,\tilde{y}_i)}, \quad \forall i \in L$

(`exp-lbl`): $E_{P(i)}[P(\tilde{y}_i|\boldsymbol{x}_i)] \geq 1 - b.$   The expectation is over the labeled set $L$, where $P(i) = 1/N_L$.

The parameter $b \in [0, 0.5)$ models the amount of label noise, and is determined from prior knowledge or can be optimized via cross-validation.

Constraint (`fix-lbl`) is written out for the binary case for simplicity. The conditionals of the labeled points are directly determined by their labels, and are treated as fixed constants. Since $b < 0.5$, the thresholded conditional classifies labeled points in the observed class. In constraint (`exp-lbl`), the conditionals for labeled points can have an average

error at most $b$, where the averaged is over all labeled points. Thus, a few points may have conditionals that deviate significantly from their observed labels, giving robustness against mislabeled points and outliers.

To obtain classification decisions, we simply choose the class with the maximum posterior $y_k = \operatorname{argmax}_y P(y|\boldsymbol{x}_k)$. Working with binary valued $P(y|\boldsymbol{x}) \in {0, 1}$ directly would yield a more difficult combinatorial optimization problem.

### 3.2 Continuous Densities

Information regularization is also computationally feasible for continuous marginal densities, known or estimated. For example, we may be given a continuous unlabeled data distribution $P(\boldsymbol{x})$ and a few discrete labeled points, and regularize across a finite set of covering regions. The conditionals are uniform inside atomic subregions (except at labeled points), requiring estimates of only a finite number of conditionals.

### 3.3 Implementation

Firstly, we choose appropriate regions forming a cover, and find the atomic subregions. The choices differ depending on whether the data is all discrete or whether continuous marginals $P(\boldsymbol{x})$ are given. Secondly, we perform a constrained optimization to find the conditionals.

If the data is all discrete, create a spherical region centered at every labeled and unlabeled point (or over some reduced set still covering all the points). We have used regions of fixed radius $R$, but the radius could also be set adaptively at each point to the distance of its $K$-nearest neighbor. The union of such regions is our cover, and we choose the radius $R$ (or $K$) large enough to create a connected cover. The cover induces a set of atomic subregions, and we merge the parameters $P(y|\boldsymbol{x})$ of points inside individual atomic subregions (atomic subregions with no observed points can be ignored). The marginal of each atomic subregion is proportional to the number of (merged) points it contains.

If continuous marginals are given, they will put probability mass in all atomic subregions where the marginal is non-zero. To avoid considering an exponential number of subregions, we can limit the overlap between the regions by creating a sparser cover.

Given the cover, we now regularize the conditionals $P(y|\boldsymbol{x})$ in the regions, according to eq. 3a. This is a convex minimization problem with a global minimum, since mutual information is convex in $P(y|\boldsymbol{x})$. It can be solved directly in the given primal form, using a quasi-Newton BFGS method. For eq. 3a, the required gradients of the constraints for the binary class ($y = \{\pm 1\}$) case (region $Q$, atomic subregion $r$) are:

$$\frac{M_Q}{V_Q} \frac{dI_Q(\boldsymbol{x}; y)}{dP(y = 1|\boldsymbol{x}_r)} = \frac{M_Q}{V_Q} P(\boldsymbol{x}_r|Q) \left( \log \frac{P(y = 1|\boldsymbol{x}_r)}{P(y = -1|\boldsymbol{x}_r)} \frac{P(y = -1|Q)}{P(y = 1|Q)} \right). \quad (4)$$

The Matlab BFGS implementation `fmincon` can solve 100 subregion problems in a few minutes.

### 3.4 Minimize Average Information

An alternative regularization criterion minimizes the average mutual information across regions. When calculating the average, we must correct for the overlaps of intersecting regions to avoid doublecounting (in contrast, the previous regularization criterion (eq. 3b) avoided doublecounting by restricting information in each region individually). The influence of a region is proportional to the probability mass $M_Q$ contained in it. However, a point $\boldsymbol{x}$ may belong to $N(\boldsymbol{x})$ regions. We define an adjusted density $P^*(\boldsymbol{x}) = P(\boldsymbol{x})/N(\boldsymbol{x})$

to calculate an adjusted probability mass $M_Q^*$ which discounts overlap. We can then minimize average mutual information according to

$$\min_{P(y|\boldsymbol{x}_k)} \sum_Q \frac{M_Q^*}{V_Q} I_Q(\boldsymbol{x};y) \tag{5a}$$

$$\text{s.t.} \quad P(y|\boldsymbol{x}_k) = \delta(y,\tilde{y}_k) \qquad\qquad \forall k \in L \tag{5b}$$

$$0 \leq P(y|\boldsymbol{x}_k) \leq 1, \quad \textstyle\sum_y P(y|\boldsymbol{x}_k) = 1 \quad \forall k \in L \cup U, \ \forall y. \tag{5c}$$

with similar necessary adjustments to incorporate noisy labels.

### 3.4.1 Limiting Behavior

The above average information criterion is a discrete version of a continuous regularization criterion. In the limit of a large number of small regions in the cover (where the spacing of the regions vanishes relative to their size), we obtain a well-defined regularization criterion resulting in continuous $P(y|\boldsymbol{x})$:

$$\min_{\substack{P(y|\boldsymbol{x}) \ \text{s.t.} \\ P(\tilde{y}_k|x_k)=\delta(y,\tilde{y}_k) \ \forall k \in L}} \int \sum_y P(x_0) P(y|x_0) \left. \left| \frac{\mathrm{d}\log P(y|x)}{\mathrm{d}x} \right|^2 \right|_{x_0} \mathrm{d}x_0. \tag{6}$$

The regularizer can also be seen as the average Fisher information (see section 2.2). More generally, we can formulate the regularization problem as a Tikhonov regularization, where the loss is the negative log-probability of labels:

$$\min_{P(y|x)} \frac{1}{N_L} \sum_{k \in L} -\log P(\tilde{y}_k|x_k) + \lambda \int \sum_y P(x_0) P(y|x_0) \left. \left| \frac{\mathrm{d}\log P(y|x)}{\mathrm{d}x} \right|^2 \right|_{x_0} \mathrm{d}x_0. \tag{7}$$

### 3.4.2 Differential Equation Characterizing the Solution

The optimization problem (eq. 6) can be solved using calculus of variations. Consider the one-dimensional binary class case and write the problem as
$\min_{P(y=1|x)} \int f\big(x, P(y=1|x), P'(y=1|x)\big) \, \mathrm{d}x$ where $f(\cdot) = P(x)P'(y=1|x)^2/[P(y=1|x)(1-P(y=1|x))]$. Necessary conditions for the solution $P(y=1|x)$ are provided by the Euler-Lagrange equations [6]

$$\frac{\partial f}{\partial P(y=1|x)} - \frac{\mathrm{d}}{\mathrm{d}x} \frac{\partial f}{\partial P'(y=1|x)} = 0 \quad \forall x. \tag{8}$$

(natural boundary conditions apply since we can assume $P(x) = 0$ and $P'(y|x) = 0$ at the boundary of the domain $\mathcal{X}$). After substituting $f$ and simplifying we have

$$P''(y=1|x) = \frac{P'(y=1|x)^2(1-2P(y=1|x))}{2P(y=1|x)(1-P(y=1|x))} - \frac{P'(x)P'(y=1|x)}{P(x)}. \tag{9}$$

This differential equation governs the solution and we solve it numerically. The labeled points provide boundary conditions, e.g. $P(y = \tilde{y}_k|x_k) = 1 - b$ for some small fixed $b \geq 0$. We must search for initial values of $P'(\tilde{y}_k|x_k)$ to match the boundary conditions of $P(\tilde{y}_k|x_k)$. The solution is continuous and piecewise differentiable.

## 4 Results and Discussion

We have experimentally studied the behavior of the regularizer with different marginal densities $P(\boldsymbol{x})$. Figure 3 shows the one-dimensional case with a continuous marginal density

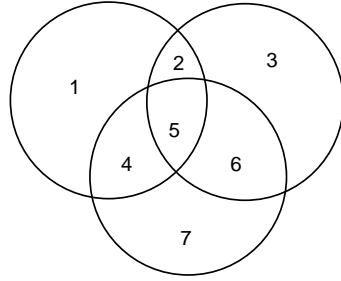 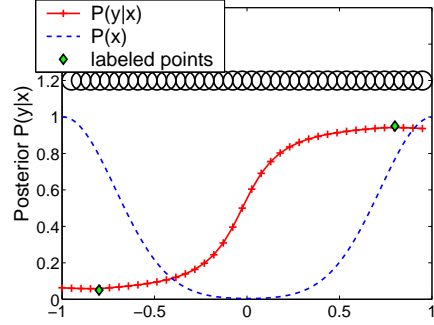

Figure 2: (Left) Three intersecting regions, and their atomic subregions (numbered). $P(y|\boldsymbol{x})$ for unlabeled points will be constant in atomic subregions.

Figure 3: (Right) The conditional (solid line) for a continuous marginal $P(\boldsymbol{x})$ (dotted line) consisting of a mixture of two continuous Gaussian and two labeled points at ($x$=-0.8,$y$=-1) and ($x$=0.8,$y$=1). The row of circles at the top depicts the region structure used (a rendering of overlapping one-dimensional intervals.)

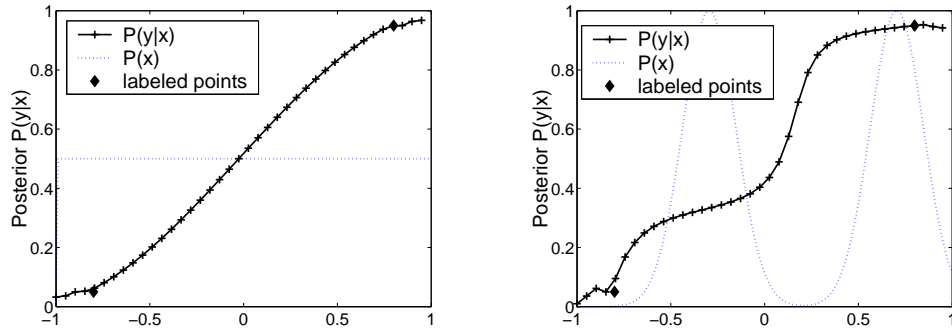

Figure 4: Conditionals (solid lines) for two continuous marginals (dotted lines) plus two labeled points. Left: the marginal is uniform, and the conditional approaches a straight line. Right: the marginal is a mixture of two Gaussians (with lower variance and shifted compared to Figure 3.) The conditional changes slowly in regions of high density.

(mixture of two Gaussians), and two discrete labeled points. We choose $N_Q$=40 regions centered at uniform intervals of $[-1, 1]$, overlapping each other half-way, creating $N_Q + 1$ atomic subregions. There are two labeled points. We show the solution attained by minimizing the maximum information (eq. 3a), and using the (`fix-lbl`) constraint with label noise $b = 0.05$.

The conditional varies smoothly between the labeled points of opposite classes. Note the dependence on the marginal density $P(\boldsymbol{x})$. The conditional is smoother in high-density regions, and changes more rapidly in low-density regions, as expected. Figure 4 shows more examples, and Figure 5 illustrates solutions obtained via the differential equation (eq. 6).

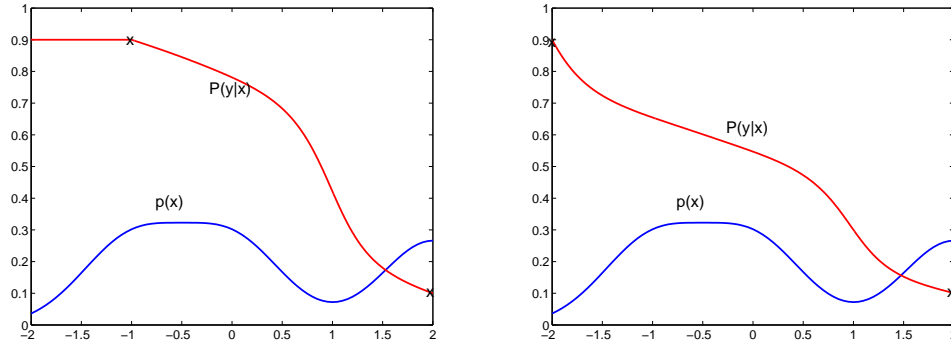

Figure 5: Conditionals for two other continuous marginals plus two labeled points (marked as crosses and located at $x$=-1, 2 in the left figure and $x$=-2, 2 in the right), solved via the differential equation (eq. 6). The conditionals are continuous but non-differentiable at the two labeled points (marked as crosses).

## 5   Conclusion

We have presented an information theoretic regularization framework for combining conditional and marginal densities in a semi-supervised estimation setting. The framework admits both discrete and continuous (known or estimated) densities. The tractability is largely a function of the number of nonempty intersections of chosen covering regions.

The principle extends beyond the presented scope. It provides flexible means of tailoring the regularizer to particular needs. The shape and structure of the regions give direct ways of imposing relations between particular variables or values of those variables. The regions can be easily defined on low-dimensional data manifolds.

In future work we will try the regularizer on large high-dimensional datasets and explore theoretical connections to network information theory.

**Acknowledgements**

The authors gratefully acknowledge support from Nippon Telegraph & Telephone (NTT) and NSF ITR grant IIS-0085836. Tommi Jaakkola also acknowledges support from the Sloan Foundation in the form of the Sloan Research Fellowship. Martin Szummer would like to thank Thomas Minka for valuable comments.

## References

[1] Tommi Jaakkola, Marina Meila, and Tony Jebara. Maximum entropy discrimination. Technical Report AITR-1668, Mass. Inst. of Technology AI lab, 1999. `http://www.ai.mit.edu/`.

[2] Naftali Tishby and Noam Slonim. Data clustering by markovian relaxation and the information bottleneck method. In *Advances in Neural Information Processing Systems (NIPS)*, volume 13, pages 640–646. MIT Press, 2001.

[3] Stephen Roberts, C. Holmes, and D. Denison. Minimum-entropy data partitioning using reversible jump Markov chain Monte Carlo. *IEEE Trans. Pattern Analysis and Mach. Intell. (PAMI)*, 23(8):909–914, 2001.

[4] Matthias Seeger. Input-dependent regularization of conditional density models. Unpublished. `http://www.dai.ed.ac.uk/homes/seeger/`, 2001.

[5] Thomas Cover and Joy Thomas. *Elements of Information Theory*. Wiley, 1991.

[6] Robert Weinstock. *Calculus of Variations*. Dover, 1974.
